# Exact differential equation population dynamics for Integrate–and–Fire neurons

**Julian Eggert** *
HONDA R&D Europe (Deutschland) GmbH
Future Technology Research
Carl-Legien-Straße 30
63073 Offenbach/Main, Germany
*julian.eggert@hre-ftr.f.rd.honda.co.jp*

**Berthold Bäuml**
Institut für Robotik und Mechatronik
Deutsches Zentrum für Luft und Raumfahrt (DLR)
Oberpfaffenhofen
*Berthold.Baeuml@dlr.de*

## Abstract

Mesoscopical, mathematical descriptions of dynamics of populations of spiking neurons are getting increasingly important for the understanding of large-scale processes in the brain using simulations. In our previous work, integral equation formulations for population dynamics have been derived for a special type of spiking neurons. For Integrate–and–Fire type neurons, these formulations were only approximately correct. Here, we derive a mathematically compact, exact population dynamics formulation for Integrate–and–Fire type neurons. It can be shown quantitatively in simulations that the numerical correspondence with microscopically modeled neuronal populations is excellent.

## 1  Introduction and motivation

The goal of the population dynamics approach is to model the time course of the collective activity of entire populations of functionally and dynamically similar neurons in a compact way, using a higher descriptional level than that of single neurons and spikes. The usual observable at the level of neuronal populations is the *population-averaged* instantaneous firing rate $A(t)$, with $A(t)\Delta t$ being the number of neurons in the population that release a spike in an interval $[t, t+\Delta t)$. Population dynamics are formulated in such a way, that they match quantitatively the time course of a given $A(t)$, either gained experimentally or by microscopical, detailed simulation.

At least three main reasons can be formulated which underline the importance of the population dynamics approach for computational neuroscience. First, it enables the simulation of extensive networks involving a massive number of neurons

and connections, which is typically the case when dealing with biologically realistic functional models that go beyond the single neuron level. Second, it increases the analytical understanding of large-scale neuronal dynamics, opening the way towards better control and predictive capabilities when dealing with large networks. Third, it enables a systematic embedding of the numerous neuronal models operating at different descriptional scales into a generalized theoretic framework, explaining the relationships, dependencies and derivations of the respective models.

Early efforts on population dynamics approaches date back as early as 1972, to the work of Wilson and Cowan [8] and Knight [4], which laid the basis for all current population-averaged graded-response models (see e.g. [6] for modeling work using these models). More recently, population-based approaches for spiking neurons were developed, mainly by Gerstner [3, 2] and Knight [5]. In our own previous work [1], we have developed a theoretical framework which enables to systematize and simulate a wide range of models for population-based dynamics. It was shown that the equations of the framework produce results that agree quantitatively well with detailed simulations using spiking neurons, so that they can be used for realistic simulations involving networks with large numbers of spiking neurons. Nevertheless, for neuronal populations composed of Integrate–and–Fire (I&F) neurons, this framework was only correct in an approximation. In this paper, we derive the exact population dynamics formulation for I&F neurons. This is achieved by reducing the I&F population dynamics to a point process and by taking advantage of the particular properties of I&F neurons.

## 2 Background: Integrate–and–Fire dynamics

### 2.1 Differential form

We start with the standard Integrate–and–Fire (I&F) model in form of the well-known differential equation [7]

$$\frac{\mathrm{d}v_i}{\mathrm{d}t}(t) = -\frac{[v_i(t) - v^{\mathrm{Rest}}]}{\tau} + j_i(t) \qquad , \tag{1}$$

which describes the dynamics of the membrane potential $v_i$ of a neuron $i$ that is modeled as a single compartment with RC circuit characteristics. The membrane relaxation time is in this case $\tau = RC$ with R being the membrane resistance and C the membrane capacitance. The resting potential $v^{\mathrm{Rest}}$ is the stationary potential that is approached in the no-input case. The input arriving from other neurons is described in form of a current $j_i$.

In addition to eq. (1), which describes the *integrate* part of the I&F model, the neuronal dynamics are completed by a nonlinear step. Every time the membrane potential $v_i$ reaches a fixed threshold $\theta$ from below, $v_i$ is lowered by a fixed amount $\Delta > 0$, and from the new value of the membrane potential integration according to eq. (1) starts again.

$$v_i(t) \to v_i(t) = v_i(t) - \Delta \qquad \text{if } v_i(t) = \theta \text{ (from below)} . \tag{2}$$

At the same time, it is said that the release of a spike occurred (i.e., the neuron fired), and the time $t_i^* = t$ of this singular event is stored. Here $t_i^*$ indicates the time of the *most recent spike*. Storing all the last firing times, we gain the sequence of spikes $\{t_i^f\}$ (spike ordering index $f$, neuronal index $i$).

## 2.2 Integral form

Now we look at the single neuron in a neuronal compound. We assume that the input current contribution $j_i$ from presynaptic spiking neurons can be described using the presynaptic spike times $t_j^f$, a response-function $\xi$ and a connection weight $w_{i,j}$

$$j_i(t) = \sum_j w_{i,j} \sum_f \xi(t - t_j^f) \qquad . \tag{3}$$

Integrating the I&F equation (1) beginning at the last spiking time $t_i^*$, which determines the initial condition by $v_i(t_i^*) = v_i(t_i^* - 0) - \Delta$, where $v_i(t_i^* - 0)$ is the membrane potential just before the neuron spikes, we get [1]

$$v_i(t) = v^{\text{Rest}} + \hat{\eta}(t - t_i^*) + \sum_j w_{i,j} \sum_f \alpha(t - t_i^*; t - t_j^f) \, , \tag{4}$$

with the *refractory function*

$$\hat{\eta}(s) = - \left( v^{\text{Rest}} - v_i(t_i^*) \right) e^{-s/\tau} \tag{5}$$

and the *alpha-function*

$$\alpha(s; s') = \int_{s'-s}^{s'} ds'' \, e^{-[s'-s'']/\tau} \, \xi(s'') \, . \tag{6}$$

If we start the integration at the time $t_i^{**}$ of the spike before the last spike, then for $t_i^{**} \le t < t_i^*$ the membrane potential is given by an expression like eq. (4), where $t_i^*$ is replaced by $t_i^{**}$. Especially we can now express $v(t_i^* - 0)$ and therefore the initial condition for an integration starting at $t_i^*$ in terms of $t_i^{**}$ and $v(t_i^{**} - 0)$. In this way, we can proceed repetitively and move back into the past. After some simple algebra this results in

$$v_i(t) = v^{\text{Rest}} + \underbrace{\sum_f \eta(t - t_i^f)}_{v_i^{\text{ref}}(t)} + \underbrace{\sum_j w_{i,j} \sum_f \alpha(\infty; t - t_j^f)}_{v_i^{\text{syn}}(t)} \, , \tag{7}$$

with a refractory function wich differs in the scale factor from that in eq. (5)

$$\eta(s) = -\Delta \, e^{-s/\tau} \, . \tag{8}$$

The components $v_i^{\text{ref}}(t)$ and $v_i^{\text{syn}}(t)$ to the membrane potential indicate refractory and synaptic components to the neuron $i$, respectively, as normally used in the Spike–Response–Model (SRM) notation [2].

Both equations (4) and (7) formulate the neuronal dynamics using a refractory component, which depends on the own spike releases of a neuron, and a synaptic component, which comprises the integrated input contribution to the membrane potential by arrival of spikes from other neurons [2]. The synaptic component is based on the alpha-function characteristic of isolated arriving spikes, with an increase of the membrane potential after spike arrival and a subsequent exponential decrease.

The comparison of the equivalent expressions eq. (4) and eq. (7) reveals an interesting property of the I&F model. They look very similar, but in eq. (4), the refractory component depends only on the time elapsed since the last spike (thus reflecting a renewal property, sometimes also called a short term memory for refractory properties), whereas in eq. (7), it depends on a sum of the contributions of all past spikes. The simpler form of the refractory contribution in eq. (4) is achieved at the cost of an alpha-function that now depends on the time $t - t_i^*$ elapsed since the last *own* spike in addition to the times $t - t_j^f$ elapsed since the release of spikes at the presynaptic neurons $j$ that provide input to $i$. In eq. (7), we have a more complex refractory contribution, but an alpha-function that does not depend on the last own spike time any more.

### 2.3  Probabilistic spike release

Probabilistic firing is introduced into the I&F model eq. (4) resp. (7) by using threshold noise. The spike release of each neuron is controlled by a hazard function $\lambda(v)$, so that

$$\lambda(v)\mathrm{d}t = \text{Prob. that a neuron with membrane potential } v \text{ spikes in } [t, t + \mathrm{d}t) \quad . \tag{9}$$

When a neuron spikes, we proceed as usual: The membrane potential is reset by a fixed amount $\Delta$ and the I&F dynamics continues.

## 3  Population dynamics

### 3.1  Density description

Descriptions of neuronal populations usually assume a neuronal density function $\rho(t; \mathbf{X})$ which depends on the state variables $\mathbf{X}$ of the neurons. The density quantifies the likelihood that a neuron picked out of the population will be found in the vicinity of the point $\mathbf{X}$ in state space,

$$\rho(t; \mathbf{X})\,\mathrm{d}\mathbf{X} = \text{Portion of neurons at time } t \text{ with state in } [\mathbf{X}, \mathbf{X} + \mathrm{d}\mathbf{X}] \quad . \tag{10}$$

If we know $\rho(t; \mathbf{X})$, the population activity $A(t)$ can be easily calculated. Using the hazard function $\lambda(t; \mathbf{X})$, the instantaneous population activity (spikes per time) can be calculated by computing the spike release averaged over the population,

$$A(t) = \int \mathrm{d}\mathbf{X}\, \lambda(t; \mathbf{X})\, \rho(t; \mathbf{X}) \quad . \tag{11}$$

The population *dynamics* is then given by the time course of the neuronal density function $\rho(t; \mathbf{X})$, which changes because each neuron evolves according to its own internal dynamics, e.g. after a spike release and the subsequent reset of the membrane potential.

The main challenge for the formulation of a population dynamics resides in selecting a low-dimensional state space [for an easy calculation of $A(t)$] and a suitable form for $\frac{\partial}{\partial t}\rho(t; \mathbf{X})$.

As an example, for the population dynamics for I&F neurons it would be straightforward to use the membrane potential $v$ from eq. (1) as the state variable $\mathbf{X}$. But this leads to a complicated density dynamics, because the dynamics for $v(t)$ consist of a continuous (differential equation (1)) and a discrete part (spike generation). Therefore, here we concentrate on an alternative description that allows a compact formulation of the desired I&F density dynamics.

## 3.2   Exact population dynamics for I&F neurons

Which is the best state space for a population dynamics of I&F neurons? For the formulation of a population dynamics, it is usually assumed that the *synaptic* contributions to the membrane potential are identical for all neurons. This is the case if we group all neurons of the same dynamical type and with identical connectivity patterns into one population. That is, we say that neurons $i$ and $i'$ belong to the same population if $w_{i,j} = w_{i',j}$ for all $j$ (for simulations of realistic networks of spiking neurons, this will of course never be exactly the case, but it is reasonable to assume that a grouping of neurons into populations can be achieved to a good approximation).

In our case, looking at eq. (4), we see that, since $\alpha(s, s')$ depends on $s = t - t_i^*$ and therefore on the own last spike time, the synaptic contribution to the membrane potential differs according to the state of the neuron. Thus we regard eq. (7). Here, we see that for identical connectivity patterns $w_{i,j}$, the synaptic contributions are the same for all neurons, because $\alpha(\infty, s')$ does not depend on the own spike time any more. Which are then the state variables of eq. (1) for the density description? We see that, for a fixed synaptic contribution, the membrane potential $v_i$ is fully determined by the set of the own past spiking times $\{t_i^f\}$. But this would mean an infinite-dimensional density for the state description of a population, and, accordingly, a computationally overly expensive calculation of the population activity $A(t)$ according to eq. (11).

To avoid this we take advantage of a particular property of the I&F model. According to eq. (8), the single spike refractory contributions $\eta(s)$ are exponential. Since any sum of exponential functions with common relaxation constant $\tau$ can be again expressed as as an exponential function with the same $\tau$, we can write instead of $v_i^{\mathrm{ref}}(t)$ from eq. (7)

$$v_i^{\mathrm{ref}}(t) = \eta_i^* \eta(t - t_i^*) \; . \tag{12}$$

Now the membrane potential $v_i(t)$ only depends on the time of the last own spike $t_i^*$ *and* the refractory contribution amplitude modulation factor at the last spike $\eta_i^*$. That is, we have transferred the effect of all spikes previous to the last one into $\eta_i^*$. In addition, we have to care about updating of $t_i^*$ and $\eta_i^*$ when a neuron spikes so that we get [3]

$$\begin{aligned}
\eta_i^* &\rightarrow & \eta_i^* &= 1 + \eta_i^* e^{-(t - t_i^*)/\tau} \; , \\
t_i^* &\rightarrow & t_i^* &= t \; .
\end{aligned} \tag{13}$$

The effect of taking into account more than the most recent spike $t_i^*$ in the refractory component $v_i^{\mathrm{ref}}(t)$ leads to a modulation factor $\eta_i^*$ greater than 1, in particular if spikes come in a rapid succession so that refractory contributions can accumulate.

Instead of using a modulation factor $\eta_i^*$ the effect of previous spikes can also be taken into account by introducing an *effective* last spiking time $\hat{t}_i^*$.

$$v_i^{\mathrm{ref}}(t) = \eta(t - \hat{t}_i^*) = \eta_i^* \eta(t - t_i^*) \; , \tag{14}$$

where $\hat{t}_i^*$ and $\eta_i^*$ are connected by

$$\hat{t}_i^* = t_i^* + \tau \ln \eta_i^* \; . \tag{15}$$

The effect of $\hat{t}^*$ is sort of funny. Because of $\eta_i^* \geq 1$ it holds for the effective last spiking time $\hat{t}_i^* \geq t_i^*$. This means, that, while at a given time $t$ it is allways $t_i^* \leq t$, it happens that $\hat{t}_i^* \geq t$, meaning the neurons behave *as if* they would spike in the *future*.

For the membrane potential we get now instead of eq. (7)

$$v_i(t) = v^{\text{Rest}} + \eta(t - \hat{t}_i^*) + \sum_j w_{i,j} \sum_f \alpha(\infty; t - t_j^f) \tag{16}$$

and for the update rule for the effective last spiking time $\hat{t}_i^*$ follows

$$\hat{t}_i^* \to \hat{t}_i^* = f(t, \hat{t}_i^*) , \tag{17}$$

with

$$f(t, \hat{t}^*) = t + \tau \ln\left[1 + e^{-(t - \hat{t}^*)/\tau}\right] . \tag{18}$$

Therefore we can regard the dimensionality of the state space of the I&F dynamics as *1-dimensional* in the description of eq. (16). The dynamics of the single I&F neurons now turns out to be very simple: Calculate the membrane potential $v_i(t)$ using eq. (16) together with the state variable $\hat{t}_i^*$, and check if $v_i(t)$ exceeds the threshold. If not, move forward in time and calculate again. If the membrane potential exceeds threshold, update $\hat{t}_i^*$ according to eq. (17) and then proceed with the calculation of $v_i(t)$ as normal.

Using this single neuron dynamics , we can now proceed to gain a population dynamics using a density $\rho(t; \hat{t}^*)$. The time $t$ is here the explicit time dependence, whereas $\hat{t}^*$ denote the state variable of the population. By fixing $\hat{t}^*$ and the synaptic contribution $v^{\text{syn}}(t)$ to the membrane potential, the state of a neuron is fully determined and the hazard function can be written as $\lambda[v^{\text{syn}}(t); \hat{t}^*]$.

The dynamics of the density $\rho(t; \hat{t}^*)$ is then calculated as follows. Changes of $\rho(t; \hat{t}^*)$ occur when neurons spike and $\hat{t}^*$ is updated according to eq. (17). The hazard function controls the spike release, and, therefore, the change of $\rho(t; \hat{t}^*)$. For chosen state variables, $\rho(t; \hat{t}^*)$ *decreases* due to spiking of the neurons with the fixed $\hat{t}^*$, and *increases* because neurons with other $\hat{t}'^*$ spike and get updated in just that way that after updating their state variable falls around $\hat{t}^*$. This occurs according to the reemplacement rule eq. (17) when

$$f(t, \hat{t}'^*) = \hat{t}^* . \tag{19}$$

Taking all together the dynamics of the density $\rho(t; \hat{t}^*)$ is given by

$$\frac{\partial}{\partial t}\rho(t; \hat{t}^*) = \overbrace{-\lambda[v^{\text{syn}}(t); \hat{t}^*]\rho(t; \hat{t}^*)}^{\text{decrease due to same state } \hat{t}^* \text{ spiking}} \tag{20}$$

$$+ \underbrace{\int_{-\infty}^{+\infty} \mathrm{d}\hat{t}'^* \, \delta[f(t, \hat{t}'^*) - \hat{t}^*] \, \lambda[v^{\text{syn}}(t); \hat{t}'^*] \, \rho(t; \hat{t}'^*)}_{\text{increase due to spiking of neurons with other states } \hat{t}'^*} .$$

The population activity can then be calculated using the density according to eq. (11) as follows

$$A(t) = \int_{-\infty}^{+\infty} \mathrm{d}\hat{t}^* \, \lambda[v^{\text{syn}}(t); \hat{t}^*] \, \rho(t; \hat{t}^*) . \tag{21}$$

Remark that the expression for the density dynamics (eq. 20) automatically conserves the norm of the density, so that

$$\int_{-\infty}^{+\infty} \mathrm{d}\hat{t}^* \rho(t; \hat{t}^*) = \text{const} , \tag{22}$$

which is a necessary condition because the number of neurons participating in the dynamics must remain constant.

## 4  Simulations

The dynamics of a population of I&F neurons, represented by the time course of their joint activity, can now be easily calculated in terms of the differential equation (20), if the neuronal state density of the neuronal population $\rho(t; \hat{t}^*)$ and the synaptic input $v^{\mathrm{syn}}(t)$ are known. This means that all we have to store is the density $\rho(t; \hat{t}^*)$ for past and future effective last spiking times $\hat{t}^*$ [4]. Favorably for numerical simulations, only a limited time window of $\hat{t}^*$ around the actual time $t$ is needed for the dynamics. The activity $A(t)$ only appears as an auxiliary variable that is calculated with the help of the neuronal density.

In figure 1 the simulation results for populations of of spiking neurons are shown. The neurons are uncoupled and a hazard function

$$\lambda(v) = \frac{1}{\tau_0} e^{2\beta(v-\theta)} \ , \tag{23}$$

with spike rate at threshold $1/\tau_0 = 1.0\mathrm{ms}^{-1}$, a kind of inverse temperature $\beta = 2.0$, which controls the noise level, and the threshold $\theta = 1.0$. The other parameters of the model in eq. (1) are: resting potential $v^{\mathrm{Rest}} = 0$, jump in membrane potential after spike release $\Delta = 1$ and time constant $\tau = 20\mathrm{ms}$. This parameters are chosen to be biologicaly plausible.

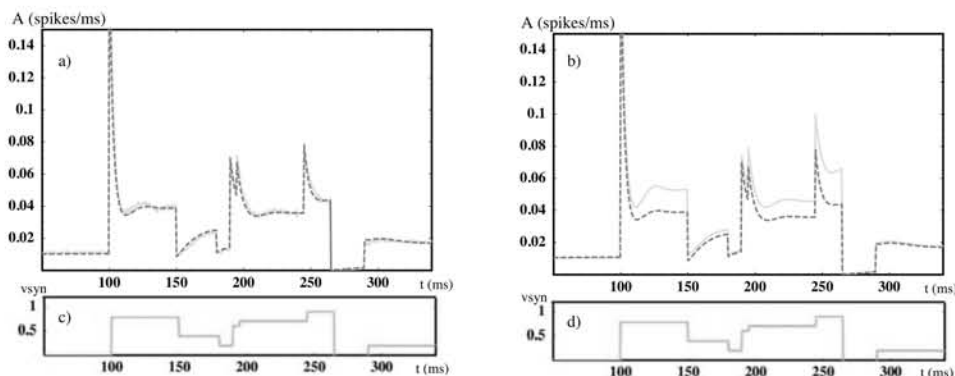

Figure 1: Activity $A(t)$ of simulated populations of neurons. The neurons are uncoupled and to each neuron the same synaptic field $v^{\mathrm{syn}}(t)$, ploted in c) and d), is applied . a) shows the activity $A(t)$ for a population of I&F neurons simulated on the one hand as $N = 10000$ single neurons (solid line) using eq. (7) and on the other hand using the density dynamics eq. (20) (dashed line). In b) the activity $A(t)$ of a population of I&F neurons (dashed line) and a population of SRM neurons with renewal (solid line) are compared. For all simulations the same parameters as specified in the text were used.

The simulations show that the density dynamics eq. (20) reproduces the activity $A(t)$ of a population of single I&F neurons almost perfect, with the exception of the noise in the single neuron simulations due to the finite size effects. This holds even for the peaks occuring at the steps of the applied synaptic field $v^{\mathrm{syn}}(t)$, although the density dynamics is entirely based on differential equations and one would therefore not expect such an excellent match for fast changes in activity.

The simulations also show that there can be a big difference between I&F and SRM neurons with renewal. Because of the accumulation of the refractory effects of all former spikes in the case of I&F neurons the activity $A(t)$ is generaly lower than for the SRM neurons with renewal and the higher the absolute actitvity level the bigger is the difference between both.

## 5   Conclusions

In this paper we derived an exact differential equation density dynamics for a population of I&F neurons starting from the microscopical equations for a single neuron. This density dynamics allows a compuationaly efficient simulation of a whole population of neurons.

In future work we want to simulate a network of connected neuronal populations. In such a network of populations (indexed e.g. by $\mathbf{x}$), a self-consistent system of differential equations based on the population's $\rho(\mathbf{x}, t; \hat{t}^*)$ and $A(\mathbf{x}, t)$ emerges if we constrain ourselves to neuronal populations connected synaptically according to the constraints given by the pool definition [2]. In this case, two neurons $i$ and $j$ belong to pools $\mathbf{x}$ and $\mathbf{y}$, if $w_{i,j} = W(\mathbf{x}, \mathbf{y})$. This allows us to write for the synaptic component of the membrane potential

$$v^{\text{syn}}(\mathbf{x}, t) = \sum_{\mathbf{y}} W(\mathbf{x}, \mathbf{y}) \int_0^\infty \mathrm{d}s' \alpha(\infty; s') A(\mathbf{y}, t - s') \qquad . \tag{24}$$

Using the alpha-function $\alpha(\infty; s')$ as introduced in (6), and a "nice" response-function $\xi$ for the input current time course after a spike, we can write eq. (24) using differential equations that use $A(\mathbf{y}, t)$ as input. This results in a system that is based entirely on differential equations and is very cheap to compute.

## Footnotes

[1] Strictly speaking, the constants $v^{\text{Rest}}$, $\tau$, $\theta$ and $\Delta$ and the function $\eta(s)$ may vary for each neuron, so that they should be written with a subindex $i$ [similarly for $\alpha(s; s')$, which may vary for each connection $j \to i$ so that we should write it with subindices $i, j$]. For the sake of clarity, we omit these indices here.

[2] So the I&F model can be formulated as a special case of the Spike–Response–Model, which *defines* the neuronal dynamics in the integral formulation.

[3]Here, the order of reemplacement matters; first we have to reemplace $\eta_i^*$, then $t_i^*$.

[4] $v^{\mathrm{syn}}(t)$ only appears as a scalar in the dynamics, so that no integration over time takes place here.

## References

[1] J. Eggert and J.L. van Hemmen. Modeling neuronal assemblies: Theory and implementation. *Neural Computation*, 13(9):1923–1974, 2001.

[2] W. Gerstner. Population dynamics of spiking neurons: Fast transients, asynchronous states and locking. *Neural Computation*, 12:43–89, 2000.

[3] W. Gerstner and J. L. van Hemmen. Associative memory in a network of 'spiking' neurons. *Network*, 3:139–164, 1992.

[4] B. W. Knight. Dynamics of encoding in a populations of neurons. *J. Gen. Physiology*, 59:734–766, 1972.

[5] B. W. Knight. Dynamics of Encoding in Neuron Populations: Some General Mathematical Features. *Neural Comput.*, 12:473–518, 2000.

[6] Z. Li. A neural model of contour integration in the primary visual cortex. *Neural Comput.*, 10(4):903–940, 1998.

[7] H. C. Tuckwell. *Introduction to Theoretical Neurobiology*. Cambridge University Press, Cambridge, 1988.

[8] H. R. Wilson and J. D. Cowan. Excitatory and inhibitory interactions in localized populations of model neurons. *Biophys. J.*, 12:1–24, 1972.
